# Automated Hierarchy Discovery for Planning in Partially Observable Environments

**Laurent Charlin & Pascal Poupart**
David R. Cheriton School of Computer Science
Faculty of Mathematics
University of Waterloo
Waterloo, Ontario
{lcharlin,ppoupart}@cs.uwaterloo.ca

**Romy Shioda**
Dept of Combinatorics and Optimization
Faculty of Mathematics
University of Waterloo
Waterloo, Ontario
rshioda@math.uwaterloo.ca

## Abstract

Planning in partially observable domains is a notoriously difficult problem. However, in many real-world scenarios, planning can be simplified by decomposing the task into a hierarchy of smaller planning problems. Several approaches have been proposed to optimize a policy that decomposes according to a hierarchy specified a priori. In this paper, we investigate the problem of automatically discovering the hierarchy. More precisely, we frame the optimization of a hierarchical policy as a non-convex optimization problem that can be solved with general non-linear solvers, a mixed-integer non-linear approximation or a form of bounded hierarchical policy iteration. By encoding the hierarchical structure as variables of the optimization problem, we can automatically discover a hierarchy. Our method is flexible enough to allow any parts of the hierarchy to be specified based on prior knowledge while letting the optimization discover the unknown parts. It can also discover hierarchical policies, including *recursive* policies, that are more compact (potentially infinitely fewer parameters) and often easier to understand given the decomposition induced by the hierarchy.

## 1 Introduction

Planning in partially observable domains is a notoriously difficult problem. However, in many real-world scenarios, planning can be simplified by decomposing the task into a hierarchy of smaller planning problems. Such decompositions can be exploited in planning to temporally abstract sub-policies into *macro* actions (a.k.a. options). Pineau et al. [17], Theocharous et al. [22], and Hansen and Zhou [10] proposed various algorithms that speed up planning in partially observable domains by exploiting the decompositions induced by a hierarchy. However these approaches assume that a policy hierarchy is specified by the user, so an important question arises: how can we automate the discovery of a policy hierarchy? In fully observable domains, there exists a large body of work on hierarchical Markov decision processes and reinforcement learning [6, 21, 7, 15] and several hierarchy discovery techniques have been proposed [23, 13, 11, 20]. However those techniques rely on the assumption that states are fully observable to detect abstractions and subgoals, which prevents their use in partially observable domains.

We propose to frame hierarchy and policy discovery as an optimization problem with variables corresponding to the hierarchy and policy parameters. We present an approach that searches in the space of hierarchical controllers [10] for a good hierarchical policy. The search leads to a difficult non-convex optimization problem that we tackle using three approaches: generic non-linear solvers, a mixed-integer non-linear programming approximation or an alternating optimization technique that can be thought as a form of hierarchical bounded policy iteration. We also generalize Hansen and Zhou's hierarchical controllers [10] to allow *recursive* controllers. These are controllers that

may recursively call themselves, with the ability of representing policies with a finite number of parameters that would otherwise require infinitely many parameters. Recursive policies are likely to arise in language processing tasks such as dialogue management and text generation due to the recursive nature of language models.

## 2   Finite State Controllers

We first review partially observable Markov decision processes (POMDPs) (Sect. 2.1), which is the framework used throughout the paper for planning in partially observable domains. Then we review how to represent POMDP policies as finite state controllers (Sect. 2.2) as well as some algorithms to optimize controllers of a fixed size (Sect. 2.3).

### 2.1   POMDPs

POMDPs have emerged as a popular framework for planning in partially observable domains [12]. A POMDP is formally defined by a tuple $(S, O, A, T, Z, R, \gamma)$ where $S$ is the set of states, $O$ is the set of observations, $A$ is the set of actions, $T(s', s, a) = \Pr(s'|s, a)$ is the transition function, $Z(o, s', a) = \Pr(o|s', a)$ is the observation function, $R(s, a) = r$ is the reward function and $\gamma \in [0, 1)$ is the discount factor. It will be useful to view $\gamma$ as a termination probability. This will allow us to absorb $\gamma$ into the transition probabilities by defining discounted transition probabilities: $\Pr_\gamma(s'|s, a) = \Pr(s'|s, a)\gamma$. Given a POMDP, the goal is to find a course of action that maximizes expected total rewards. To select actions, the system can only use the information available in the past actions and observations. Thus we define a policy $\pi$ as a mapping from histories of past actions and observations to actions. Since histories may become arbitrarily long, we can alternatively define policies as mappings from beliefs to actions (i.e., $\pi(b) = a$). A belief $b(s) = \Pr(s)$ is a probability distribution over states, taking into account the information provided by past actions and observations. Given a belief $b$, after executing $a$ and receiving $o$, we can compute an updated belief $b^{a,o}$ using Bayes' theorem: $b^{a,o}(s) = kb(s)\Pr(s'|s, a)\Pr(o|s'a)$. Here $k$ is a normalization constant. The value $V^\pi$ of policy $\pi$ when starting with belief $b$ is measured by the expected sum of the future rewards: $V^\pi(b) = \sum_t R(b_t, \pi(b_t))$, where $R(b, a) = \sum_s b(s)R(s, a)$. An optimal policy $\pi^*$ is a policy with the highest value $V^*$ for all beliefs (i.e., $V^*(b) \geq V^\pi(b) \forall b, \pi$). The optimal value function also satisfies Bellman's equation: $V^*(b) = \max_a \left( R(b, a) + \gamma \Pr(o|b, a)V^*(b^{a,o}) \right)$, where $\Pr(o|b, a) = \sum_{s,s'} b(s)\Pr(s'|s, a)\Pr(o|s', a)$.

### 2.2   Policy Representation

A convenient representation for an important class of policies is that of finite state controllers [9]. A finite state controller consists of a finite state automaton $(N, E)$ with a set $N$ of nodes and a set $E$ of directed edges Each node $n$ has one outgoing edge per observation. A controller encodes a policy $\pi = (\alpha, \beta)$ by mapping each node to an action (i.e., $\alpha(n) = a$) and each edge (referred by its observation label $o$ and its parent node $n$) to a successor node (i.e., $\beta(n, o) = n'$). At runtime, the policy encoded by a controller is executed by doing the action $a_t = \alpha(n_t)$ associated with the node $n_t$ traversed at time step $t$ and following the edge labelled with observation $o_t$ to reach the next node $n_{t+1} = \beta(n_t, o_t)$.

Stochastic controllers [18] can also be used to represent stochastic policies by redefining $\alpha$ and $\beta$ as distributions over actions and successor nodes. More precisely, let $\Pr_\alpha(a|n)$ be the distribution from which an action $a$ is sampled in node $n$ and let $\Pr_\beta(n'|n, a, o)$ be the distribution from which the successor node $n'$ is sampled after executing $a$ and receiving $o$ in node $n$. The value of a controller is computed by solving the following system of linear equations:

$$V_n^\pi(s) = \sum_a \Pr_\alpha(a|n)[R(s, a) + \sum_{s',o,n'} \Pr_\gamma(s'|s, a)\Pr(o|s', a)\Pr_\beta(n'|n, a, o)V_{n'}^\pi(s')] \; \forall n, s \quad (1)$$

While there always exists an optimal policy representable by a deterministic controller, this controller may have a very large (possibly infinite) number of nodes. Given time and memory constraints, it is common practice to search for the best controller with a bounded number of nodes [18]. However, when the number of nodes is fixed, the best controller is not necessarily deterministic. This explains why searching in the space of stochastic controllers may be advantageous.

Table 1: Quadratically constrained optimization program for bounded stochastic controllers [1].

$$\max_{x,y} \quad \sum_s b_o(s) \underbrace{V_{n_o}(s)}_{y}$$

$$\text{s.t.} \quad \underbrace{V_n(s)}_{y} = \sum_{a,n'} \left[ \underbrace{\Pr(a,n'|n,o_k)}_{x} R(s,a) + \sum_{s',o} \Pr_\gamma(s'|s,a) \Pr(o|s',a) \underbrace{\Pr(n',a|n,o)}_{x} \underbrace{V_{n'}(s')}_{y} \right] \quad \forall n,s$$

$$\underbrace{\Pr(n',a|n,o)}_{x} \geq 0 \quad \forall n',a,n,o \qquad \sum_{n',a} \underbrace{\Pr(n',a|n,o)}_{x} = 1 \quad \forall n,o$$

$$\sum_{n'} \underbrace{\Pr(n',a|n,o)}_{x} = \sum_{n'} \underbrace{\Pr(n',a|n,o_k)}_{x} \quad \forall a,n,o$$

## 2.3 Optimization of Stochastic Controllers

The optimization of a stochastic controller with a fixed number of nodes can be formulated as a quadratically constrained optimization problem (QCOP) [1]. The idea is to maximize $V^\pi$ by varying the controller parameters $\Pr_\alpha$ and $\Pr_\beta$. Table 1 describes the optimization problem with $V_n(s)$ and the joint distribution $\Pr(n',a|n,o) = \Pr_\alpha(n|a)\Pr_\beta(n'|n,a,o)$ as variables. The first set of constraints corresponds to those of Eq. 1 while the remaining constraints ensure that $\Pr(n',a|n,o)$ is a proper distribution and that $\sum_{n'}' \Pr(n',a|n,o) = \Pr_\alpha(a|n)\forall o$. This optimization program is non-convex due to the first set of constraints. Hence, existing techniques can at best guarantee convergence to a local optimum. Several techniques have been tried including gradient ascent [14], stochastic local search [3], bounded policy iteration (BPI) [18] and a general non-linear solver called SNOPT (based on sequential quadratic programming) [1, 8]. Empirically, biased-BPI (version of BPI that biases its search to the belief region reachable from a given initial belief state) and SNOPT have been shown to outperform the other approaches on some benchmark problems [19, 1]. We quickly review BPI since it will be extended in Section 3.2 to optimize hierarchical controllers. BPI alternates between policy evaluation and policy improvement. Given a policy with fixed parameters $\Pr(a,n'|n,o)$, policy evaluation solves the linear system in Eq 1 to find $V_n(s)$ for all $n,s$. Policy improvement can be viewed as a linear simplification of the program in Table 1 achieved by fixing $V_{n'}(s')$ in the right hand side of the first set of constraints. Policy improvement is achieved by optimizing the controller parameters $\Pr(n',a|n,o)$ and the value $V_n(s)$ on the left hand side.[1]

## 3 Hierarchical controllers

Hansen and Zhou [10] recently proposed hierarchical finite-state controllers as a simple and intuitive way of encoding hierarchical policies. A hierarchical controller consists of a set of nodes and edges as in a flat controller, however some nodes may be abstract, corresponding to sub-controllers themselves. As with flat controllers, concrete nodes are parameterized with an action mapping $\alpha$ and edges outgoing concrete nodes are parameterized by a successor node mapping $\beta$. In contrast, abstract nodes are parameterized by a child node mapping indicating in which child node the sub-controller should start. Hansen and Zhou consider two schemes for the edges outgoing abstract nodes: either there is a single outgoing edge labelled with a null observation or there is one edge per terminal node of the subcontroller labelled with an abstract observation identifying the node in which the subcontroller terminated.

Subcontrollers encode full POMDP policies with the addition of a termination condition. In fully observable domains, it is customary to stop the subcontroller once a goal state (from a predefined set of terminal states) is reached. This strategy cannot work in partially observable domains, so Hansen and Zhou propose to terminate a subcontroller when an end node (from a predefined set of terminal nodes) is reached. Since the decision to reach a terminal node is made according to the successor node mapping $\beta$, the timing for returning control is implicitly optimized. Hansen and

Zhou propose to use $|A|$ terminal nodes, each mapped to a different action. Terminal nodes do not have any outgoing edges nor any action mapping since they already have an action assigned.

The hierarchy of the controller is assumed to be finite and specified by the programmer. Subcontrollers are optimized in isolation in a bottom up fashion. Subcontrollers at the bottom level are made up only of concrete nodes and therefore can be optimized as usual using any controller optimization technique. Controllers at other levels may contain abstract nodes for which we have to define the reward function and the transition probabilities. Recall that abstract nodes are not mapped to concrete actions, but rather to children nodes. Hence, the immediate reward of an abstract node $\bar{n}$ corresponds to the value $V_{\alpha(\bar{n})}(s)$ of its child node $\alpha(\bar{n})$. Similarly, the probability of reaching state $s'$ after executing the subcontroller of an abstract node $\bar{n}$ corresponds to the probability $\Pr(s_{end}|s, \alpha(\bar{n}))$ of terminating the subcontroller in $s_{end}$ when starting in $s$ at child node $\alpha(\bar{n})$. This transition probability can be computed by solving the following linear system:

$$\Pr(s_{end}|s, n) = \begin{cases} 1 \text{ when } n \text{ is a terminal node and } s = s_{end} \\ 0 \text{ when } n \text{ is a terminal node and } s \neq s_{end} \\ \sum_{o,s'} \Pr(s'|s, \alpha(n)) \Pr(o|s', \alpha(n)) \Pr(s_{end}|s', \beta(n,o)) \quad \text{otherwise} \end{cases} \quad (2)$$

Subcontrollers with abstract actions correspond to partially observable semi-Markov decision processes (POSMDPs) since the duration of each abstract action may vary. The duration of an action is important to determine the amount by which future rewards should be discounted. Hansen and Zhou propose to use the mean duration to determine the amount of discounting, however this approach does not work. In particular, abstract actions with non-zero probability of never terminating have an infinite mean duration. Instead, we propose to absorb the discount factor into the transition distribution (i.e., $\Pr_{\gamma}(s'|s, a) = \gamma \Pr(s'|s, a)$). This avoids all issues related to discounting and allows us to solve POSMDPs with the same algorithms as POMDPs. Hence, given the abstract reward function $R(s, \alpha(\bar{n})) = V_{\alpha(\bar{n})}(s)$ and the abstract transition function $\Pr_{\gamma}(s'|s, \alpha(\bar{n}))$ obtained by solving the linear system in Eq. 2, we have a POSMDP which can be optimized using any POMDP optimization technique (as long as the discount factor is absorbed into the transition function).

Hansen's hierarchical controllers have two limitations: the hierarchy must have a finite number of levels and it must be specified by hand. In the next section we describe recursive controllers which may have infinitely many levels. We also describe an algorithm to discover a suitable hierarchy by simultaneously optimizing the controller parameters and hierarchy.

## 3.1 Recursive Controllers

In some domains, policies are naturally *recursive* in the sense that they decompose into subpolicies that may call themselves. This is often the case in language processing tasks since language models such as probabilistic context-free grammars are composed of recursive rules. Recent work in dialogue management uses POMDPs to make high level discourse decisions [24]. Assuming POMDP dialogue management eventually handles decisions at the sentence level, recursive policies will naturally arise. Similarly, language generation with POMDPs would naturally lead to recursive policies that reflect the recursive nature of language models.

We now propose several modifications to Hansen and Zhou's hierarchical controllers that simplify things while allowing recursive controllers. First, the subcontrollers of abstract nodes may be composed of any node (including the parent node itself) and transitions can be made to any node anywhere (whether concrete or abstract). This allows recursive controllers and smaller controllers since nodes may be shared across levels. Second, we use a single terminal node that has no action nor any outer edge. It is a virtual node simply used to signal the termination of a subcontroller. Third, while abstract nodes lead to the execution of a subcontroller, they are also associated with an action. This action is executed upon termination of the subcontroller. Hence, the actions that were associated with the terminal nodes in Hansen and Zhou's proposal are associated with the abstract nodes in our proposal. This allows a uniform parameterization of actions for all nodes while reducing the number of terminal nodes to 1. Fourth, the outer edges of abstract nodes are labelled with regular observations since an observation will be made following the execution of the action of an abstract node. Finally, to circumvent all issues related to discounting, we absorb the discount factor into the transition probabilities (i.e., $\Pr_{\gamma}(s'|s, a)$).

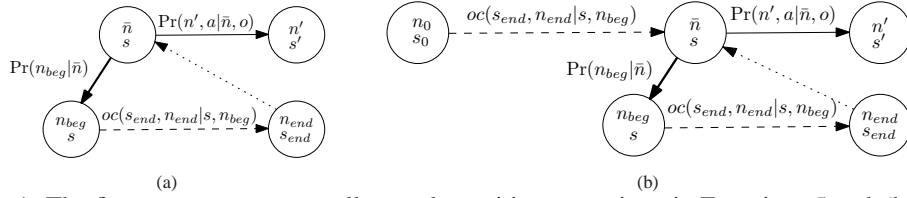

<center>(a)</center> <center>(b)</center>

Figure 1: The figures represent controllers and transitions as written in Equations 5 and 6b. Alongside the directed edges we've indicated the equivalent part of the equations which they correspond to.

## 3.2 Hierarchy and Policy Optimization

We formulate the search for a good stochastic recursive controller, including the automated hierarchy discovery, as an optimization problem (see Table 2). The global maximum of this optimization problem corresponds to the optimal policy (and hierarchy) for a fixed set $N$ of concrete nodes $n$ and a fixed set $\bar{N}$ of abstract nodes $\bar{n}$. The variables consist of the value function $V_n(s)$, the policy parameters $\Pr(n', a|n, o)$, the (stochastic) child node mapping $\Pr(n'|\bar{n})$ for each abstract node $\bar{n}$ and the occupancy frequency $oc(n, s|n_0, s_0)$ of each $(n, s)$-pair when starting in $(n_0, s_0)$. The objective (Eq. 3) is the expected value $\sum_s b_0(s) V_{n_0}(s)$ of starting the controller in node $n_0$ with initial belief $b_0$. The constraints in Equations 4 and 5 respectively indicate the expected value of concrete and abstract nodes. The expected value of an abstract node corresponds to the sum of three terms: the expected value $V_{n_{beg}}(s)$ of its subcontroller given by its child node $n_{beg}$, the reward $R(s_{end}, a_{\bar{n}})$ immediately after the termination of the subcontroller and the future rewards $V_n(s')$. Figure 1a illustrates graphically the relationship between the variables in Equation 5. Circles are state-node pairs labelled by their expected value. Edges indicate single transitions (solid line), sequences of transitions (dashed line) or the beginning/termination of a subcontroller (bold/dotted line). Edges are labelled with the corresponding transition probability variables.

Note that the reward $R(s_{end}, a_{\bar{n}})$ depends on the state $s_{end}$ in which the subcontroller terminates. Hence we need to compute the probability that the last state visited in the subcontroller is $s_{end}$. This probability is given by the occupancy frequency $oc(s_{end}, n_{end}|s, n_{beg})$, which is recursively defined in Eq. 6 in terms of a preceding state-node pair which may be concrete (6a) or abstract (6b). Figure 1b illustrates graphically the relationship between the variables in Eq. 6b. Eq. 7 prevents infinite loops (without any action execution) in the child node mappings. The *label* function refers to the labelling of all abstract nodes, which induces an ordering on the abstract nodes. Only the nodes labelled with numbers larger than the label of an abstract node can be children of that abstract node. This constraint ensures that chains of child node mappings have a finite length, eventually reaching a concrete node where an action is executed. Constraints, like the ones in Table 1, are also needed to guarantee that the policy parameters and the child node mappings are proper distributions.

## 3.3 Algorithms

Since the problem in Table 2 has non-convex (quartic) constraints in Eq. 5 and 6, it is difficult to solve. We consider three approaches inspired from the techniques for non-hierarchical controllers:

**Non-convex optimization:** Use a general non-linear solver, such as SNOPT, to directly tackle the optimization problem in Table 2. This is the most convenient approach, however a globally optimal solution may not be found due to the non-convex nature of the problem.

**Mixed-Integer Non-Linear Programming (MINLP):** We restrict $\Pr(n', a|n, o)$ and $\Pr(n_{beg}|\bar{n})$ to be binary (i.e., in $\{0, 1\}$). Since the optimal controller is often near deterministic in practice, this restriction tends to have a negligible effect on the value of the optimal controller. The problem is still non-convex but can be tackled with a mixed-integer non-linear solver such as MINLP_BB [2].

**Bounded Hierarchical Policy Iteration (BHPI):** We alternate between (i) solving a simplified version of the optimization where some variables are fixed and (ii) updating the values of the fixed variables. More precisely, we fix $V_{n'}(s')$ in Eq. 5 and $oc(s, \bar{n}|s_0, n_0)$ in Eq. 6. As a result, Eq. 5 and 6 are now cubic, involving products of variables that include a single continuous variable. This per-

Table 2: Non-convex quarticly constrained optimization problem for hierarchy and policy discovery in bounded stochastic recursive controllers.

$$\max_{w,x,y,z} \quad \sum_{s \in S} b_0(s) \underbrace{V_{n_0}(s)}_{y} \tag{3}$$

$$\text{s.t.} \quad \underbrace{V_n(s)}_{y} = \sum_{a,n'} \left[ \underbrace{\Pr(n',a|n,o_k)}_{x} R(s,a) + \sum_{s',o} \Pr_\gamma(s'|s,a) \Pr(o|s',a) \underbrace{\Pr(n',a|n,o)}_{x} \underbrace{V_{n'}(s')}_{y} \right] \quad \forall s,n \tag{4}$$

$$\underbrace{V_{\bar{n}}(s)}_{y} = \sum_{n_{beg}} \underbrace{\Pr(n_{beg}|\bar{n})}_{z} \left[ \underbrace{V_{n_{beg}}(s)}_{y} + \sum_{s_{end},a,n'} \underbrace{oc(s_{end},n_{end}|s,n_{beg})}_{w} \left[ \underbrace{\Pr(n',a|\bar{n},o_k)}_{x} R(s_{end},a) \right. \right.$$

$$\left. \left. + \sum_{s',o} \Pr_\gamma(s'|s_{end},a) \Pr(o|s',a) \underbrace{\Pr(n',a|\bar{n},o)}_{x} \underbrace{V_{n'}(s')}_{y} \right] \right] \quad \forall s,\bar{n} \tag{5}$$

$$\underbrace{oc(s',n'|s_0,n_0)}_{w} = \delta(s',n',s_0,n_0) + \sum_{s,o,a} \left[ \vphantom{\sum} \right. \tag{6}$$

$$\left. \sum_n \underbrace{oc(s,n|s_0,n_0)}_{w} \Pr_\gamma(s'|s,a) \Pr(o|s',a) \underbrace{\Pr(n',a|n,o)}_{x} \right] \qquad \left. \right\} \ n \text{ concrete } (6a)$$

$$+ \sum_{s_{end},n_{beg},\bar{n}} \underbrace{oc(s,\bar{n}|s_0,n_0)}_{w} \Pr_\gamma(s'|s_{end},a) \Pr(o|s',a)$$

$$\underbrace{oc(s_{end},n_{end}|s,n_{beg})}_{w} \underbrace{\Pr(n',a|\bar{n},o)}_{x} \underbrace{\Pr(n_{beg}|\bar{n})}_{z} \right] \quad \forall s_0,s',n_0,n' \qquad \left. \right\} \ \bar{n} \text{ abstract } (6b)$$

$$\Pr(\bar{n}'|\bar{n}) = 0 \text{ if } label(\bar{n}') \leq label(\bar{n}), \forall \bar{n}, \bar{n}' \tag{7}$$

mits the use of disjunctive programming [2] to linearize the constraints without any approximation. The idea is to replace any product $BX$ (where $B$ is binary and $X$ is continuous) by a new continuous variable $Y$ constrained by $lb_X B \leq Y \leq ub_X B$ and $X + (B-1)ub_X \leq Y \leq X + (B-1)lb_X$ where $lb_X$ and $ub_X$ are lower and upper bounds on $X$. One can verify that those additional linear constraints force $Y$ to be equal to $BX$. After applying disjunctive programming, we solve the resulting mixed-integer *linear* program (MILP) and update $V_{n'}(s')$ and $oc(s,\bar{n}|s_0,n_0)$ based on the new values for $V_n(s)$ and $oc(s',n'|s_0,n_0)$. We repeat the process until convergence or until a pre-defined time limit is reached. Although, convergence cannot be guaranteed, in practice we have found BHPI to be monotonically increasing. Note that fixing $V_{n'}(s')$ and $oc(s,\bar{n}|s_0,n_0)$ while varying the policy parameters is reminiscent of policy iteration, hence the name *bounded hierarchical policy iteration*.

### 3.4 Discussion

Discovering a hierarchy offers many advantages over previous methods that assume the hierarchy is already known. In situations where the user is unable to specify the hierarchy, our approach provides a principled way of discovering it. In situations where the user has a hierarchy in mind, it may be possible to find a better one. Note however that discovering the hierarchy while optimizing the policy is a much more difficult problem than simply optimizing the policy parameters. Additional variables (e.g., $\Pr(n',a|n,o)$ and $oc(s,n|s_0,n_0)$) must be optimized and the degree of non-linearity increases. Our approach can also be used when the hierarchy and the policy are partly known. It is fairly easy to set the variables that are known or to reduce their range by specifying upper and lower bounds. This also has the benefit of simplifying the optimization problem.

It is also interesting to note that hierarchical policies may be encoded with exponentially fewer nodes in a hierarchical controller than a flat controller. Intuitively, when a subcontroller is called by $k$ abstract nodes, this subcontroller is shared by all its abstract parents. An equivalent flat controller would have to use $k$ separate copies of the subcontroller. If a hierarchical controller has $l$ levels with subcontrollers shared by $k$ parents in each level, then the equivalent flat controller will need $O(k^l)$ copies. By allowing recursive controllers, policies may be represented even more compactly. Recursive controllers allow abstract nodes to call subcontrollers that may contain themselves. An

Table 3: Experiment results

| Problem | S | A | O | V* | Num. of Nodes | SNOPT | | BHPI | | MINLP_BB | |
|---|---|---|---|---|---|---|---|---|---|---|---|
| | | | | | | Time | V | Time | V | Time | V |
| Paint | 4 | 4 | 2 | 3.3 | 4(3/1) | 2s | 0.48 | 13s | 3.29 | <1s | 3.29 |
| Shuttle | 8 | 3 | 5 | 32.7 | 4(3/1) | 2s | 31.87 | 85s | 18.92 | 4s | 18.92 |
| | 8 | 3 | 5 | 32.7 | 6(4/2) | 6s | 31.87 | 7459s | 27.93 | 221s | 27.68 |
| | 8 | 3 | 5 | 32.7 | 7(4/3) | 26s | 31.87 | 10076s | 31.87 | N/A | – |
| | 8 | 3 | 5 | 32.7 | 9(5/4) | 1449s | 30.27 | 10518s | 3.73 | N/A | – |
| 4x4 Maze | 16 | 4 | 2 | 3.7 | 3(2/1) | 3s | 3.15 | 397s | 3.21 | 30s | 3.73 |

equivalent non-hierarchical controller would have to unroll the recursion by creating a separate copy of the subcontroller each time it is called. Since recursive controllers essentially call themselves infinitely many times, they can represent infinitely large non-recursive controllers with finitely many nodes. As a comparison, recursive controllers are to non-recursive hierarchical controllers what context-free grammars are to regular expressions. Since the leading approaches for controller optimization fix the number of nodes [18, 1], one may be able to find a much better policy by considering hierarchical recursive controllers. In addition, hierarchical controllers may be easier to understand and interpret than flat controllers given their natural decomposition into subcontrollers and their possibly smaller size.

## 4 Experiments

We report on some preliminary experiments with three toy problems (paint, shuttle and maze) from the POMDP repository[3]. We used the SNOPT package to directly solve the non-convex optimization problem in Table 2 and bounded hierarchical policy iteration (BHPI) to solve it iteratively. Table 3 reports the running time and the value of the hierarchical policies found.[4] For comparison purposes, the optimal value of each problem (copied from [4]) is reported in the column labelled by $V^*$. We optimized hierarchical controllers of two levels with a fixed number of nodes reported in the column labelled "Num. of Nodes". The numbers in parentheses indicate the number of nodes at the top level (left) and at the bottom level (right).[5] In general, SNOPT finds the optimal solution with minimal computational time. In contrast, BHPI is less robust and takes up to several orders of magnitude longer. MINLP_BB returns good solutions for the smaller problems but is unable to find feasible solutions to the larger ones. We also looked at the hierarchy discovered for each problem and verified that it made sense. In particular, the hierarchy discovered for the paint problem matches the one hand coded by Pineau in her PhD thesis [16]. Given the relatively small size of the test problems, these experiments should be viewed as a proof of concept that demonstrate the feasibility of our approach. More extensive experiments with larger problems will be necessary to demonstrate the scalability of our approach.

## 5 Conclusion & Future Work

This paper proposes the first approach for hierarchy discovery in partially observable planning problems. We model the search for a good hierarchical policy as a non-convex optimization problem with variables corresponding to the hierarchy and policy parameters. We propose to tackle the optimization problem using non-linear solvers such as SNOPT or by reformulating the problem as an approximate MINLP or as a sequence of MILPs that can be thought of as a form of hierarchical bounded policy iteration. Preliminary experiments demonstrate the feasibility of our approach, however further research is necessary to improve scalability. The approach can also be used in situations where a user would like to improve or learn part of the hierarchy. Many variables can then be set (or restricted to a smaller range) which simplifies the optimization problem and improves scalability.

We also generalize Hansen and Zhou's hierarchical controllers to *recursive* controllers. Recursive controllers can encode policies with finitely many nodes that would otherwise require infinitely large

non-recursive controllers. Further details about recursive controllers and our other contributions can be found in [5]. We plan to further investigate the use of recursive controllers in dialogue management and text generation where recursive policies are expected to naturally capture the recursive nature of language models.

**Acknowledgements:** this research was supported by the Natural Sciences and Engineering Research Council (NSERC) of Canada, the Canada Foundation for Innovation (CFI) and the Ontario Innovation Trust (OIT).

## Footnotes

[1]Note however that this optimization may decrease the value of some nodes so [18] add an additional constraint to ensure monotonic improvement by forcing $V_n(s)$ on the left hand side to be at least as high as $V_n(s)$ on the right hand side.

[2] http://www-unix.mcs.anl.gov/~leyffer/solvers.html

[3] http://pomdp.org/pomdp/examples/index.shtml

[4] N/A refers to a trial when the solver was unable to return a feasible solution to the problem.

[5] Since the problems are simple, the number of levels was restricted to two, though our approach permits any number of levels and does not require the number of levels nor the number of nodes per level to be specified.

## References

[1] C. Amato, D. Bernstein, and S. Zilberstein. Solving POMDPs using quadratically constrained linear programs. In *To appear In International Joint Conferences on Artificial Intelligence (IJCAI)*, 2007.

[2] E. Balas. Disjunctive programming. *Annals of Discrete Mathematics*, 5:3–51, 1979.

[3] D. Braziunas and C. Boutilier. Stochastic local search for POMDP controllers. In *AAAI*, pages 690–696, 2004.

[4] A. Cassandra. *Exact and approximate algorithms for partially observable Markov decision processes*. PhD thesis, Brown University, Dept. of Computer Science, 1998.

[5] L. Charlin. Automated hierarchy discovery for planning in partially observable domains. Master's thesis, University of Waterloo, 2006.

[6] T. Dietterich. Hierarchical reinforcement learning with the MAXQ value function decomposition. *JAIR*, 13:227–303, 2000.

[7] M. Ghavamzadeh and S. Mahadevan. Hierarchical policy gradient algorithms. In T. Fawcett and N. Mishra, editors, *ICML*, pages 226–233. AAAI Press, 2003.

[8] P. Gill, W. Murray, and M. Saunders. SNOPT: An SQP algorithm for large-scale constrained optimization. *SIAM Review*, 47(1):99–131, 2005.

[9] E. Hansen. An improved policy iteration algorithm for partially observable MDPs. In *NIPS*, 1998.

[10] E. Hansen and R. Zhou. Synthesis of hierarchical finite-state controllers for POMDPs. In E. Giunchiglia, N. Muscettola, and D. Nau, editors, *ICAPS*, pages 113–122. AAAI, 2003.

[11] B. Hengst. Discovering hierarchy in reinforcement learning with HEXQ. In *ICML*, pages 243–250, 2002.

[12] L. Kaelbling, M. Littman, and A. Cassandra. Planning and acting in partially observable stochastic domains. *Artificial Intelligence*, 101(1-2):99–134, 1998.

[13] A. McGovern and A. Barto. Automatic discovery of subgoals in reinforcement learning using diverse density. In *ICML*, pages 361–368, 2001.

[14] N. Meuleau, L. Peshkin, K.-E. Kim, and L. Kaelbling. Learning finite-state controllers for partially observable environments. In *UAI*, pages 427–436, 1999.

[15] R. Parr. *Hierarchical Control and learning for Markov decision processes*. PhD thesis, University of California at Berkeley, 1998.

[16] J. Pineau. *Tractable Planning Under Uncertainty: Exploiting Structure*. PhD thesis, Robotics Institute, Carnegie Mellon University, 2004.

[17] J. Pineau, G. Gordon, and S. Thrun. Policy-contingent abstraction for robust robot control. In *UAI*, pages 477–484, 2003.

[18] P. Poupart and C. Boutilier. Bounded finite state controllers. In *NIPS*, 2003.

[19] Pascal Poupart. *Exploiting Structure to efficiently solve large scale partially observable Markov decision processes*. PhD thesis, University of Toronto, 2005.

[20] M. Ryan. Using abstract models of behaviours to automatically generate reinforcement learning hierarchies. In *ICML*, pages 522–529, 2002.

[21] R. Sutton, D. Precup, and S. Singh. Between MDPs and Semi-MDPs: A framework for temporal abstraction in reinforcement learning. *Artificial Intelligence*, 112(1-2):181–211, 1999.

[22] G. Theocharous, S. Mahadevan, and L. Kaelbling. Spatial and temporal abstractions in POMDPs applied to robot navigation. Technical Report MIT-CSAIL-TR-2005-058, Computer Science and Artificial Intelligence Laboratory, MIT, 2005.

[23] S. Thrun and A. Schwartz. Finding structure in reinforcement learning. In *NIPS*, pages 385–392, 1994.

[24] J. Williams and S. Youngs. Scaling POMDPs for dialogue management with composite summary point-based value iteration (CSPBVI). In *AAAI workshop on Statistical and Empirical Methods in Spoken Dialogue Systems*, 2006.